# Context dependent amplification of both rate and event-correlation in a VLSI network of spiking neurons

**Elisabetta Chicca, Giacomo Indiveri and Rodney J. Douglas**
Institute of Neuroinformatics
University - ETH Zurich
Winterthurerstrasse 190, CH-8057 Zurich, Switzerland
`chicca,giacomo,rjd@ini.phys.ethz.ch`

## Abstract

Cooperative competitive networks are believed to play a central role in cortical processing and have been shown to exhibit a wide set of useful computational properties. We propose a VLSI implementation of a spiking cooperative competitive network and show how it can perform context dependent computation both in the mean firing rate domain and in spike timing correlation space. In the mean rate case the network amplifies the activity of neurons belonging to the selected stimulus and suppresses the activity of neurons receiving weaker stimuli. In the event correlation case, the recurrent network amplifies with a higher gain the correlation between neurons which receive highly correlated inputs while leaving the mean firing rate unaltered. We describe the network architecture and present experimental data demonstrating its context dependent computation capabilities.

## 1   Introduction

There is an increasing body of evidence supporting the hypothesis that recurrent cooperative competitive neural networks play a central role in cortical processing [1]. Anatomical studies demonstrated that the majority of synapses in the mammalian cortex originate within the cortex itself [1, 2]. Similarly, it has been shown that neurons with similar functional properties are aggregated together in *modules* or *columns* and most connections are made locally within the neighborhood of a 1 mm column [3].

From the computational point of view, recurrent cooperative competitive networks have been investigated extensively in the past [4–6]. Already in the late 70's Amari and Arbib [4] applied the concept of *dynamic neural fields*[1] [7, 8] to develop a unifying mathematical framework to study cooperative competitive neural network models based on a series of detailed models of biological systems [9–11]. In 1994, Douglas et al. [5] argued that recurrent cortical circuits restore analog signals on the basis of their connectivity patterns and produce selective neuronal responses while maintaining network stability. To support this hypothesis they proposed the *cortical amplifier*[2] model and showed that a network of cortical amplifiers performs signal restoration and noise suppression by amplifying the correlated signal in a pattern that was stored in the connectivity of the network, without amplifying the noise. In 1998, Hansel and Sompolinsky presented a detailed model for cor-

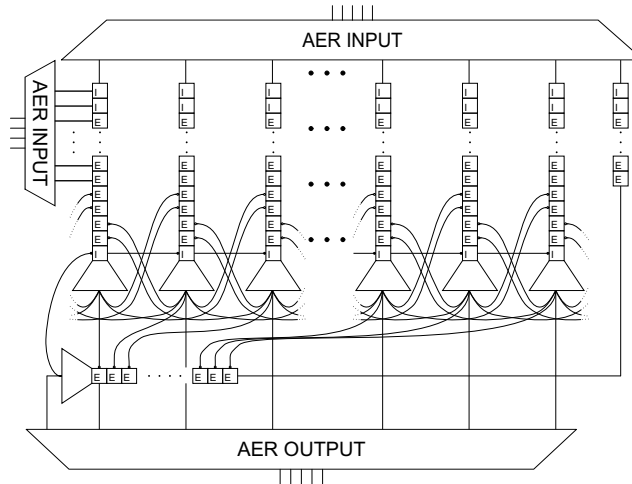

Figure 1: The chip architecture. Squares represent excitatory (E) and inhibitory (I) synapses, trapezoids represent I&F neurons. The synapse can be stimulated by external (AER) inputs and by local events. The I&F neurons can transmit their spikes off-chip and/or to the locally connected synapses (see text for details). The local connectivity implements a cooperative competitive network with first and secon dneighbors recurrent excitatory connections and global inhibition. The first and second neighbor connections of the neurons at the edges of the array are connected to pads. This allows us to leave the network open, or implement closed boundary conditions (to form a ring of neurons), using off-chip jumpers. The global inhibitory neuron (bottom left) receives excitation from all neurons in the array and its output inhibits all of them.

tical feature selectivity based on recurrent cooperative competitive networks [6] where they showed how these models can account for some of the emergent cooperative cortical properties observed in nature.

Recently it has been argued that recurrent cooperative competitive networks exhibit at the same time computational properties both in an analog way (e.g. amplification or filtering) and in a digital way (e.g. digital selection) [12]. To demonstrate the digital selection and analog amplification capabilities of these type of networks Hahnloser et al. proposed a VLSI chip in which neurons are implemented as linear threshold units and input and output signals encode mean firing rates. The recurrent connectivity of the network proposed in [12] comprises self-excitation, first and second neighbors recurrent excitatory connections, and global inhibition. We are particularly interested in the use of these types of networks in spike based multi-chip sensory systems [13] as a computational module capable of performing stimulus selection, signal restoration and noise suppression. Here we propose a spike-based VLSI neural network that allows us to explore these computational properties both in the mean rate and time domain.

The device we propose comprises a ring of excitatory Integrate-and-Fire (I&F) neurons with first and second neighbors recurrent excitatory connections and a global inhibitory neuron which receives excitation from all the neurons in the ring. In the next Section we describe the network architecture, and in Section 3 and 4 we show how this network can perform context dependent computation in the mean rate domain and in the time domain respectively.

## 2   The VLSI Spiking Cooperative Competitive Network

Several examples of VLSI competitive networks of spiking neurons have already been presented in literature [14–19]. In 1992, De Yong et al. [16] proposed a VLSI winner-take-all (WTA) spiking network consisting of 4 neurons with all-to-all inhibitory connections. In 1993, a different VLSI WTA chip comprising also 4 neurons was proposed, it used global inhibition to implement the WTA behavior [17]. More recent implementations of spiking VLSI cooperative competitive networks consist of larger arrays and show more complex behavior thanks also to more advanced VLSI processes and testing instruments currently available [14, 15, 18, 19].

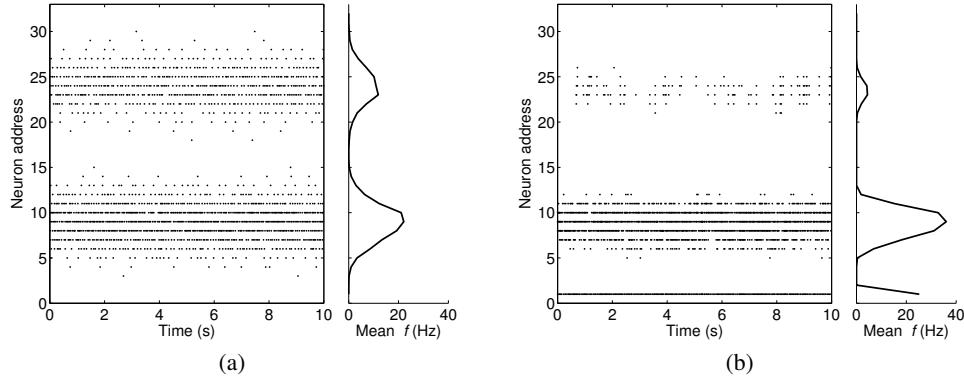

Figure 2: Raster plot for the suppression experiments. (a) Feed-forward network response. The left panel shows the raster plot of the network activity in response to two Gaussian shaped, Poisson distributed, input spike trains, with mean firing rates ranging from 0 to 120 Hz. The right panel shows the mean frequencies of the neurons ion the feed-forward network. The feed-forward network response directly reflects the applied input: on average, the neurons produce an output spike in response to about every 6 input spikes. Note how the global inhibitory neuron (address number 1) is not active. (b) Cooperative-competitive network response. The left panel shows the raster plot of the network activity in response to the same input applied to the feed-forward network. The right panel shows the mean output frequencies of the neurons in the cooperative-competitive network. The recurrent connectivity (lateral excitation and global inhibition) amplifies the activity of the neurons with highest mean output frequency and suppresses the activity of other neurons (compare with right panel of (a)).

Most of the previously proposed VLSI models focused on hard WTA behaviors (only the neuron that receives the strongest input is active). Our device allows us to explore hard and soft WTA behaviors both in the mean rate and spike timing domain. Here we explore the network's ability to perform context dependent computation using the network in the soft WTA mode.

The VLSI network we designed comprises 31 excitatory neurons and 1 global inhibitory neuron [15]. Cooperation between neurons is implemented by first and second neighbors recurrent excitatory connections. Depending on the relative strength of excitation and inhibition the network can be operated either in hard or soft WTA mode. On top of the local hardwired recurrent connectivity the network comprises 16 AER[3] synapses per neuron.

The chip was fabricated in a 0.8 $\mu m$, n-well, double metal, double poly, CMOS process using the Europractice service.

The architecture of the VLSI network of I&F neurons is shown in Fig. 1. It is a two-dimensional array containing a row of 32 neurons, each connected to a column of afferent synaptic circuits. Each column contains 14 AER excitatory synapses, 2 AER inhibitory synapses and 6 locally connected (hard-wired) synapses. The circuits implementing the chip's I&F neurons and synapses have been described in [22].

When an input address-event is received, the synapse with the corresponding row and column address is stimulated. If the input address-events routed to the synapse integrate up to the neuron's spiking threshold, then that neuron generates an output address-event which is transmitted off-chip. Arbitrary network architectures can be implemented using off-chip look-up tables and routing the chip's output address-events to one or more AER input synapses. The synapse address can belong to a different chip, therefore, arbitrary multi-chip architectures can be implemented.

Synapses with local hard-wired connectivity are used to realize the cooperative competitive network with nearest neighbor and second nearest neighbor interactions (see Fig. 1): 31 neurons of the array

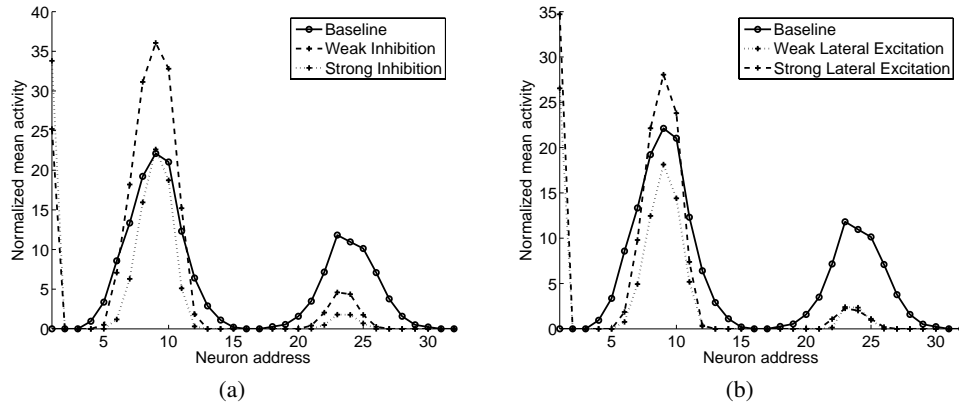

<div align="center">(a)                                               (b)</div>

Figure 3: Suppression as a function of the strength of global inhibition and lateral excitation. (a) The graph shows the mean firing rate of the neurons for three different connectivity conditions and in response to the same stimulus. The continuous line represents the baseline: the activity of the neurons in response to the external stimulus when the local connections are not active (feed-forward network). The dashed and dotted lines represent the activity of the feed-back network for weak and strong inhibition respectively and fixed lateral excitation. For weak inhibition the neurons that receive the strongest inputs amplify their activity. (b) Activity of the network for three different connectivity conditions. The continuous line represents the baseline (response of the feed-forward network to the input stimulus). The dashed and dotted lines represent the activity of the feed-back network for weak and strong lateral excitation respectively and fixed global inhibition. For strong lateral excitation the neurons that receive the highest input amplify their activity.

send their spikes to 31 local excitatory synapses on the global inhibitory neuron; the inhibitory neuron, in turn, stimulates local inhibitory synapses of the 31 excitatory neurons; each excitatory neuron stimulates its first and second neighbors on both sides using two sets of locally connected synapses. The first and second neighbor connections of the neurons at the edges of the array are connected to pads. This allows us to leave the network open, or implement closed boundary conditions (to form a ring of neurons [12]), using off-chip jumpers.

All of the synapses on the chip can be switched off. This allows us to inactivate either the local synaptic connections, or the AER ones, or to use local synapses in conjunction with the AER ones. In addition, a uniform constant DC current can be injected to all the neurons in the array thus producing a regular "spontaneous" activity throughout the whole array.

## 3   Competition in mean rate space

In our recurrent network competition is implemented using one global inhibitory neuron and co-operation using first and second nearest neighbors excitatory connections, nonetheless it performs complex non-linear operations similar to those observed in more general cooperative competitive networks. These networks, often used to model cortical feature selectivity [6, 23], are typically tested with bell-shaped inputs. Within this context we can map sensory inputs (e.g. obtained from a silicon retina, a silicon cochlea, or other AER sensory systems) onto the network's AER synapses in a way to implement different types of feature maps. For example, Chicca et al. [24] recently presented an orientation selectivity system implemented by properly mapping the activity of a silicon retina onto AER input synapses of our chip. Moreover the flexibility of the AER infrastructure, combined with the large number of externally addressable AER synapses of our VLSI device, allows us to perform cooperative competitive computation across different feature spaces in parallel.

We explored the behavior of the network using synthetic control stimuli: we stimulated the chip via its input AER synapses with Poisson distributed spike trains, using Gaussian shaped mean frequency profiles. A custom PCI-AER board [24] was used to stimulate the chip and monitor its activity.

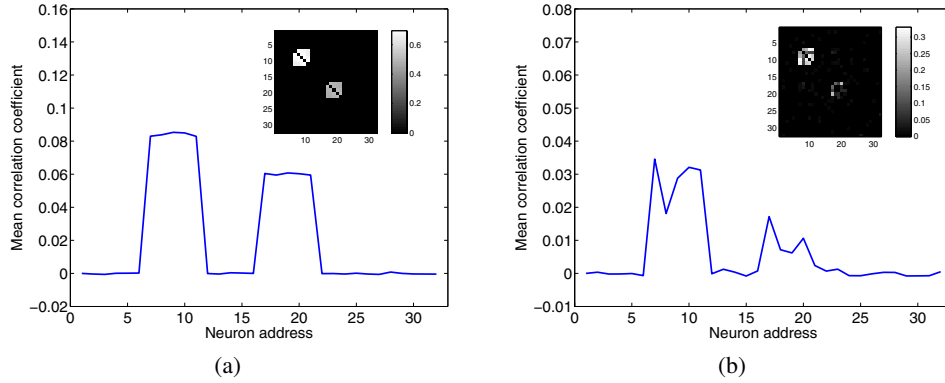

(a)                                    (b)

Figure 4: (a) Mean correlation coefficient among input spike trains used to stimulate the neurons of our network. The figure inset shows the pairwise correlations between each input source. (b) Mean correlation coefficient of output spike trains, when the cooperative-competitive connections of the network are disabled (feed-forward mode). Note the different scales on the y-axis and in the inset color bars.

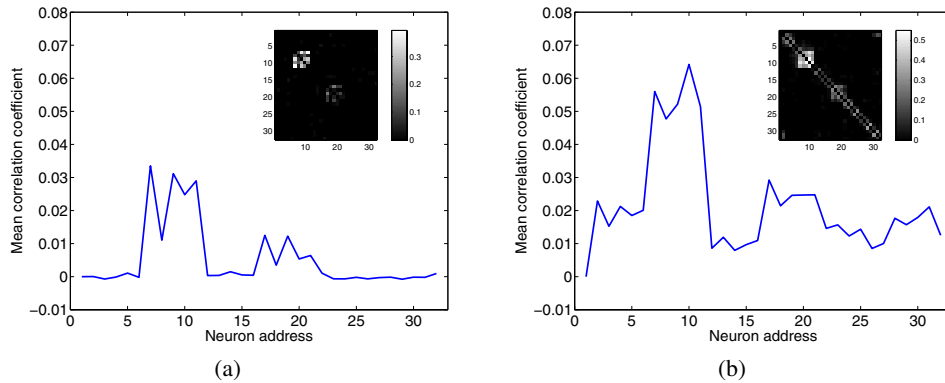

(a)                                    (b)

Figure 5: (a) Correlation coefficient of output spike trains, when only global inhibition is enabled; (b) Correlation coefficient of output spike trains when both global inhibition and local excitation are enabled.

Suppression of least effective stimuli was tested using two Gaussian shaped inputs with different amplitude (in terms of mean frequency) composed by Poisson trains of spikes. Two examples of raw data for these experiments in the feed-forward and recurrent network conditions are shown in Fig. 2(a) and 2(b) respectively. The output of the network is shown in Fig. 3(a) for two different values of the strength of global inhibition (modulated using the weight of the connection from the excitatory neurons to the global inhibitory neuron) and a fixed strength of lateral excitation. The activity of the recurrent network has to be compared with the activity of the feed-forward network ("baseline" activity plotted in Fig. 2(a) and represented by the continuous line in Fig. 3(a)) in response to the same stimulus to easily estimate the effect of the recurrent connectivity. The most active neurons cooperatively amplify their activity through lateral excitation and efficiently drive the global inhibitory neuron to suppress the activity of other neurons (dashed line in Fig. 3(a)). When the strength of global inhibition is high the amplification given by the lateral excitatory connections can be completely suppressed (dotted line in Fig. 3(a)). A similar behavior is observed when the strength of lateral excitation is modulated (see Fig. 3(b)). For strong lateral excitation (dashed line in Fig. 3(b)) amplification is observed for the neurons receiving the input with highest mean frequency and suppression of neurons stimulated by trains with lower mean frequencies occur. When lateral excitation is weak (dotted line in Fig. 3(b)), global inhibition dominates and the activity of all neurons is suppressed.

The non-linearity of this behavior is evident when we compare the effect of recurrent connectivity on the peak of the lowest hill of activity and on the side of the highest hill of activity (e.g. neuron 23 and 11 respectively, in Fig. 3(a)). In the feed-forward network (continuous line) these two neurons have a similar mean out frequency ($\sim 12\ Hz$), nevertheless the effect of recurrent connectivity on their activity is different. The activity of neuron 11 is amplified by a factor of 1.24 while the activity of neuron 23 is suppressed by a factor of 0.39 (dashed line). This difference shows that the network is able to act differently on similar mean rates depending on the spatial context, distinguishing the relevant signal from distractors and noise.

## 4   Competition in correlation space

Here we test the context-dependent computation properties of the cooperative competitive network, also in the spike-timing domain. We stimulated the neurons with correlated, Poisson distributed spike trains and analyzed the network's response properties in correlation space, as a function of its excitatory/inhibitory connection settings.

Figure 4(a) shows the mean correlation coefficient between each input spike train with the spike trains sent to all other neurons in the array. The figure inset shows the pair-wise correlation coefficient across all neuron addresses: neurons 7 through 11 have one common source of input (35Hz) and five independent sources (15Hz) for a total mean firing rate of 50Hz and a 70% correlation; neurons 17 through 21 were stimulated with one common source of input at 25Hz and five independent sources at 25HZ, for a total mean firing rate of 50Hz and a 50% correlation; all other neurons were stimulated with uncorrelated sources at 50Hz. The auto-correlation coefficients (along the diagonal in the figure's inset) are not plotted, for sake of clarity.

When used as a plain feed-forward network (with all local connections disabled), the neurons generate output spike trains that reflect the distributions of the input signal, both in the mean firing rate domain (see Fig.2(a)) and in the correlation domain (see Fig.4(b)). The lower output mean firing rates and smaller amount of correlations among output spikes are due to the integrating properties of the I&F neuron and of the AER synapses.

In Fig.5 we show the response of the network when global inhibition and recurrent local excitation are activated. Enabling only global inhibition, without recurrent excitation has no substantial effect with respect to the feed-forward case (compare Fig.5(a) with Fig.4(b)). However, when both competition and cooperation are enabled the network produces context-dependent effects in the correlation space that are equivalent to the ones observed in the mean-rate domain: the correlation among neurons that received inputs with highest correlation is amplified, with respect to the feed-forward case, while the correlation between neurons that were stimulated by weakly correlated sources is comparable to the correlation between all other neurons in the array.

Given the nature of the connectivity patterns in our chip, the correlation among neighboring neurons is increased throughout the array, independent of the input sources, hence the mean correlation coefficient is higher throughout the whole network. However, the difference in correlation between the base level and the group with highest correlation is significantly higher when cooperation and competition are enabled, with respect to the feed-forward case. At the same time, the difference in correlation between the base level and the group with lowest correlation when cooperation and competition are enabled cannot be distinguished from that of the feed-forward case. See Tab. 1 for the estimated mean and standard deviation in the four conditions.

These are preliminary experiments that provide encouraging results. We are currently in the process of designing an equivalent architecture one a new chip using an AMS $0.35\mu$m technology, with 256 neurons and 8192 synapses. We will use the new chip to perform much more thorough experiments to extend the analysis presented in this Section.

## 5   Discussion

We presented a hardware cooperative competitive network composed of spiking VLSI neurons and analog synapses, and used it to simulate in real-time network architectures similar to those studied by Amari and Arbib [4], Douglas et al. [5], Hansel and Sompolinsky [6], and Dayan and Abbott [25]. We showed how the hardware cooperative competitive network can exhibit the type of complex

Table 1: Difference in correlation between the base level and the two groups with correlated input in the feed-forward and cooperative-competitive network

|  | Feed-forward network | Cooperative competitive network |
|---|---|---|
| Highest correlation group | $0.029 \pm 0.007$ | $0.04 \pm 0.01$ |
| Lowest correlation group | $0.009 \pm 0.006$ | $0.010 \pm 0.007$ |

non-linear behaviors observed in biological neural systems. These behaviors have been extensively studied in continuous models but were never demonstrated in hardware spiking systems before. We pointed out how the recurrent network can act differently on neurons with similar activity depending on the local context (i.e. mean firing rates, or mean correlation coefficient).

In the mean rate case the network amplifies the activity of neurons belonging to the selected stimulus and suppresses the activity of neurons belonging to distractors or at noise level. This property is particular relevant in the context of signal restoration. We believe that this is one of the mechanisms used by biological systems to perform highly reliable computation restoring signals on the basis of cooperative-competitive interaction among elementary units of recurrent networks and hence on the basis of the *context* of the signal.

In the mean correlation coefficient case, the recurrent network amplifies more efficiently the correlation between neurons which receive highly correlated inputs while keeping the average mean firing rate constant. This result supports the idea that correlation can be viewed as an additional coding dimension for building internal representations [26].

**Acknowledgments**

This work was supported in part by the EU grants ALAVLSI (IST-2001-38099) and DAISY (FP6-2005-015803), and in part by the Swiss National Science Foundation (PMPD2-110298/1).

## Footnotes

[1]In the *dynamic neural fields* approach neural networks are described as a continuous medium rather than a set of discrete neurons. A differential equation describes the activation of the *neural tissue* at different position in the neural network.

[2]The *cortical amplifier* consists of a population of identical neurons, connected to each other with the same excitatory synaptic strength, sharing a common inhibitory feedback and the same input.

[3]In the Address Event Representation (AER) input and output spikes are real-time digital events that carry analog information structure [20]. An asynchronous communication protocol based on the AER is the most efficient for signal transmission across neuromorphic devices [21].

# References

[1] R. J. Douglas and K. A. C. Martin. Neural circuits of the neocortex. *Annual Review of Neuroscience*, 27:419–51, 2004.

[2] T. Binzegger, R. J. Douglas, and K. Martin. A quantitative map of the circuit of cat primary visual cortex. *Journal of Neuroscience*, 24(39):8441–53, 1994.

[3] E. R. Kandel, J.H. Schwartz, and T. M. Jessell. *Principles of Neural Science*. Mc Graw Hill, 2000.

[4] S. Amari and M. A. Arbib. Competition and cooperation in neural nets. In J. Metzler, editor, *Systems Neuroscience*, pages 119–65. Academic Press, 1977.

[5] R. J. Douglas, M. A. Mahowald, and K. A. C. Martin. Hybrid analog-digital architectures for neuromorphic systems. In *Proc. IEEE World Congress on Computational Intelligence*, volume 3, pages 1848–1853. IEEE, 1994.

[6] D. Hansel and H. Somplinsky. *Methods in Neuronal Modeling*, chapter Modeling Feature Selectivity in Local Cortical Circuits, pages 499–567. MIT Press, Cambridge, Massachusetts, 1998.

[7] S. Amari. Dynamics of pattern formation in lateral-inhibition type neural fields. *Biological Cybernetics*, 27:77–87, 1977.

[8] W. Erlhagen and G. Schöner. Dynamic field theory of movement preparation. *Psychological Review*, 109:545–572, 2002.

[9] P. Dev. Perception of depth surfaces in random–dot stereograms: a neural model. *International Journal of Man–Machine Studies*, 7:511–28, 1975.

[10] R. L. Didday. A model of visuomotor mechanisms in the frog optic tectum. *Mathematical Biosciences*, 30:169–80, 1976.

[11] W. L. Kilmer, W. S. McCulloch, and J. Blum. A model of the vertebrate central command system. *International Journal of Man-Machine Studies*, 1:279–309, 1969.

[12] R. Hahnloser, R. Sarpeshkar, M. Mahowald, R. J. Douglas, and S. Seung. Digital selection and analog amplification co-exist in an electronic circuit inspired by neocortex. *Nature*, 405(6789):947–951, 2000.

[13] R. Serrano-Gotarredona, M. Oster, P. Lichtsteiner, A. Linares-Barranco, R. Paz-Vicente, F. Gómez-Rodríguez, H. Kolle Riis, T. Delbrück, S. C. Liu, S. Zahnd, A. M. Whatley, R. J. Douglas, P. Häfliger, G. Jimenez-Moreno, A. Civit, T. Serrano-Gotarredona, A. Acosta-Jiménez, and B. Linares-Barranco. AER building blocks for multi-layer multi-chip neuromorphic vision systems. In S. Becker, S. Thrun, and K. Obermayer, editors, *Advances in Neural Information Processing Systems*, volume 15. MIT Press, Dec 2005.

[14] J.P. Abrahamsen, P. Hafliger, and T.S. Lande. A time domain winner-take-all network of integrate-and-fire neurons. In *2004 IEEE International Symposium on Circuits and Systems*, volume 5, pages V–361 – V–364, May 2004.

[15] E. Chicca, G. Indiveri, and R. J. Douglas. An event based VLSI network of integrate-and-fire neurons. In *Proceedings of IEEE International Symposium on Circuits and Systems*, pages V-357–V-360. IEEE, 2004.

[16] M. R. DeYong, R. L. Findley, and C. Fields. The design, fabrication, and test of a new VLSI hybrid analog-digital neural processing element. *IEEE Transactions on Neural Networks*, 3(3):363–74, 1992.

[17] P. Hylander, J. Meador, and E. Frie. VLSI implementaion of pulse coded winner take all networks. In *Proceedings of the 36th Midwest Symposium on Circuits and Systems*, volume 1, pages 758–761, 1993.

[18] G. Indiveri, R. Mürer, and J. Kramer. Active vision using an analog VLSI model of selective attention. *IEEE Transactions on Circuits and Systems II*, 48(5):492–500, May 2001.

[19] M. Oster and S.-C. Liu. A winner-take-all spiking network with spiking inputs. In *11th IEEE International Conference on Electronics, Circuits and Systems (ICECS 2004)*, 2004.

[20] K. A. Boahen. *Retinomorphic Vision Systems: Reverse Engineering the Vertebrate Retina*. Ph.D. thesis, California Institute of Technology, Pasadena, CA, 1997.

[21] E. Culurciello and A. G. Andreou. A comparative study of access topologies for chip-level address-event communication channels. *IEEE Transactions on Neural Networks*, 14(5):1266–77, September 2003.

[22] E. Chicca, D. Badoni, V. Dante, M. D'Andreagiovanni, G. Salina, S. Fusi, and P. Del Giudice. A VLSI recurrent network of integrate–and–fire neurons connected by plastic synapses with long term memory. *IEEE Transactions on Neural Networks*, 14(5):1297–1307, September 2003.

[23] R. Ben-Yishai, R. Lev Bar-Or, and H. Sompolinsky. Theory of orientation tuning in visual cortex. *Proceedings of the National Academy of Sciences of the USA*, 92(9):3844–3848, April 1995.

[24] E. Chicca, A. M. Whatley, V. Dante, P. Lichtsteiner, T. Delbruck, P. Del Giudice, R. J. Douglas, and G. Indiveri. A multi-chip pulse-based neuromorphic infrastructure and its application to a model of orientation selectivity. *IEEE Transactions on Circuits and Systems I, Regular Papers*, 2006. (In press).

[25] P. Dayan and F. Abbott. *Theoretical Neuroscience: Computational and Mathematical Modeling of Neural Systems*. MIT Press, 2001.

[26] E. Salinas and T. J. Sejnowski. Correlated neuronal activity and the flow of neural information. *Nature Reviews Neuroscience*, 2:539–550, 2001.
